# Worst-Case Analysis of Selective Sampling for Linear-Threshold Algorithms[*]

**Nicolò Cesa-Bianchi**
DSI, University of Milan
*cesa-bianchi@dsi.unimi.it*

**Claudio Gentile**
Università dell'Insubria
*gentile@dsi.unimi.it*

**Luca Zaniboni**
DTI, University of Milan
*zaniboni@dti.unimi.it*

## Abstract

We provide a worst-case analysis of selective sampling algorithms for learning linear threshold functions. The algorithms considered in this paper are Perceptron-like algorithms, i.e., algorithms which can be efficiently run in any reproducing kernel Hilbert space. Our algorithms exploit a simple margin-based randomized rule to decide whether to query the current label. We obtain selective sampling algorithms achieving on average the same bounds as those proven for their deterministic counterparts, but using much fewer labels. We complement our theoretical findings with an empirical comparison on two text categorization tasks. The outcome of these experiments is largely predicted by our theoretical results: Our selective sampling algorithms tend to perform as good as the algorithms receiving the true label after each classification, while observing in practice substantially fewer labels.

## 1 Introduction

In this paper, we consider learning binary classification tasks with partially labelled data via selective sampling. A selective sampling algorithm (e.g., [3, 12, 7] and references therein) is an on-line learning algorithm that receives a sequence of unlabelled instances, and decides whether or not to query the label of the current instance based on instances and labels observed so far. The idea is to let the algorithm determine which labels are most useful to its inference mechanism, so that redundant examples can be discarded on the fly and labels can be saved.

The overall goal of selective sampling is to fit real-world scenarios where labels are scarce or expensive. As a by now classical example, in a web-searching task, collecting web pages is a fairly automated process, but assigning them a label (a set of *topics*) often requires time-consuming and costly human expertise. In these cases, it is clearly important to devise learning algorithms having the ability to exploit the label information as much as possible. Furthermore, when we consider kernel-based algorithms [23, 9, 21], saving labels directly implies saving support vectors in the currently built hypothesis, which, in turn, implies saving running time in both training and test phases.

Many algorithms have been proposed in the literature to cope with the broad task of learning with partially labelled data, working under both probabilistic and worst-case assumptions, for either on-line or batch settings. These range from active learning algorithms [8, 22],

---

[*]The authors gratefully acknowledge partial support by the PASCAL Network of Excellence under EC grant no. 506778. This publication only reflects the authors' views.

to the query-by-committee algorithm [12], to the adversarial "apple tasting" and label-efficient algorithms investigated in [16] and [17, 6], respectively. In this paper we present a worst-case analysis of two Perceptron-like selective sampling algorithms. Our analysis relies on and contributes to a well-established way of studying linear-threshold algorithms within the mistake bound model of on-line learning (e.g., [18, 15, 11, 13, 14, 5]). We show how to turn the standard versions of the (first-order) Perceptron algorithm [20] and the second-order Perceptron algorithm [5] into selective sampling algorithms exploiting a randomized margin-based criterion (inspired by [6]) to select labels, while preserving in expectation the same mistake bounds.

In a sense, this line of research complements an earlier work on selective sampling [7], where a second-order kind of algorithm was analyzed under precise stochastic assumptions about the way data are generated. This is exactly what we face in this paper: we avoid any assumption whatsoever on the data-generating process, but we are still able to prove meaningful statements about the label efficiency features of our algorithms.

In order to give some empirical evidence for our analysis, we made some experiments on two medium-size text categorization tasks. These experiments confirm our theoretical results, and show the effectiveness of our margin-based label selection rule.

## 2 Preliminaries, notation

An example is a pair $(\boldsymbol{x}, y)$, where $\boldsymbol{x} \in \mathbb{R}^n$ is an *instance* vector and $y \in \{-1, +1\}$ is the associated binary label. A training set $S$ is any finite sequence of examples $S = (\boldsymbol{x}_1, y_1), \ldots, (\boldsymbol{x}_T, y_T) \in (\mathbb{R}^n \times \{-1, +1\})^T$. We say that $S$ is linearly separable if there exists a vector $\boldsymbol{u} \in \mathbb{R}^n$ such that $y_t \boldsymbol{u}^\top \boldsymbol{x}_t > 0$ for $t = 1, \ldots, T$.

We consider the following selective sampling variant of a standard on-line learning model (e.g., [18, 24, 19, 15] and references therein). This variant has been investigated in [6] for a version of Littlestone's Winnow algorithm [18, 15]. Learning proceeds on-line in a sequence of *trials*. In the generic trial $t$ the algorithm receives instance $\boldsymbol{x}_t$ from the environment, outputs a prediction $\hat{y}_t \in \{-1, +1\}$ about the label $y_t$ associated with $\boldsymbol{x}_t$, and *decides* whether or not to query the label $y_t$. No matter what the algorithm decides, we say that the algorithm has made a prediction *mistake* if $\hat{y}_t \neq y_t$. We measure the performance of the algorithm by the total number of mistakes it makes on $S$ (including the trials where the true label remains hidden). Given a *comparison class* of predictors, the goal of the algorithm is to bound the amount by which this total number of mistakes differs, on an *arbitrary* sequence $S$, from some measure of the performance of the best predictor in hindsight within the comparison class. Since we are dealing with (zero-threshold) linear-threshold algorithms, it is natural to assume the comparison class be the set of all (zero-threshold) linear-threshold predictors, i.e., all (possibly normalized) vectors $\boldsymbol{u} \in \mathbb{R}^n$. Given a margin value $\gamma > 0$, we measure the performance of $\boldsymbol{u}$ on $S$ by its cumulative *hinge loss*[1] [11, 13] $\sum_{t=1}^T D_\gamma(\boldsymbol{u}; (\boldsymbol{x}_t, y_t))$, where $D_\gamma(\boldsymbol{u}; (\boldsymbol{x}_t, y_t)) = \max\{0, \gamma - y_t \boldsymbol{u}^\top \boldsymbol{x}_t\}$.

Broadly speaking, the goal of the selective sampling algorithm is to achieve the best bound on the number of mistakes with as few queried labels as possible. As in [6], our algorithms exploit a margin-based randomized rule to decide which labels to query. Thus, our mistake bounds are actually worst-case over the training sequence and average-case over the internal randomization of the algorithms. All expectations occurring in this paper are w.r.t. this randomization.

## 3 The algorithms and their analysis

As a simple example, we start by turning the classical Perceptron algorithm [20] into a worst-case selective sampling algorithm. The algorithm, described in Figure 1, has a real

Figure 1: The selective sampling (first-order) Perceptron algorithm.

parameter $b > 0$ which might be viewed as a noise parameter, ruling the extent to which a linear threshold model fits the data at hand. The algorithm maintains a vector $\boldsymbol{v} \in \mathbb{R}^n$ (whose initial value is zero). In each trial $t$ the algorithm observes an instance vector $\boldsymbol{x}_t \in \mathbb{R}^n$ and predicts the binary label $y_t$ through the sign of the margin value $r_t = \boldsymbol{v}_{k-1}^\top \hat{\boldsymbol{x}}_t$. Then the algorithm decides whether to ask for the label $y_t$ through a simple randomized rule: a coin with bias $b/(b + |r_t|)$ is flipped; if the coin turns up heads ($Z_t = 1$ in Figure 1) then the label $y_t$ is revealed. Moreover, on a prediction mistake ($\hat{y}_t \neq y_t$) the algorithm updates vector $\boldsymbol{v}_k$ according to the usual Perceptron additive rule. On the other hand, if either the coin turns up tails or $\hat{y}_t = y_t$ no update takes place. Notice that $k$ is incremented only when an update occurs. Thus, at the end of trial $t$, subscript $k$ counts the number of updates made so far (plus one). In the following theorem we prove that our selective sampling version of the Perceptron algorithm can achieve, in expectation, the same mistake bound as the standard Perceptron's using fewer labels. See Remark 1 for a discussion.

**Theorem 1** *Let* $S = ((\boldsymbol{x}_1, y_1), (\boldsymbol{x}_2, y_2), \ldots, (\boldsymbol{x}_T, y_T)) \in (\mathbb{R}^n \times \{-1, +1\})^T$ *be any sequence of examples and* $\mathcal{U}_T$ *be the (random) set of* update *trials for the algorithm in Figure 1 (i.e, the set of trials* $t \leq T$ *such that* $\hat{y}_t \neq y_t$ *and* $Z_t = 1$*). Then the expected number of mistakes made by the algorithm in Figure 1 is upper bounded by*

$$\inf_{\gamma > 0} \inf_{\boldsymbol{u} \in \mathbb{R}^n} \left( \frac{2b+1}{2b} \mathbb{E}\left[ \sum_{t \in \mathcal{U}_T} \frac{1}{\gamma} D_\gamma(\boldsymbol{u}; (\hat{\boldsymbol{x}}_t, y_t)) \right] + \frac{(2b+1)^2}{8b} \frac{||\boldsymbol{u}||^2}{\gamma^2} \right) .$$

*The expected number of labels queried by the algorithm is equal to* $\sum_{t=1}^T \mathbb{E}\left[ \frac{b}{b + |r_t|} \right]$.

*Proof.* Let $M_t$ be the Bernoulli variable which is one iff $\hat{y}_t \neq y_t$ and denote by $k(t)$ the value of the update counter $k$ in trial $t$ *just before* the update $k \leftarrow k + 1$. Our goal is then to bound $\mathbb{E}\left[ \sum_{t=1}^T M_t \right]$ from above. Consider the case when trial $t$ is such that $M_t Z_t = 1$. Then one can verify by direct inspection that choosing $r_t = \boldsymbol{v}_{k(t-1)}^\top \hat{\boldsymbol{x}}_t$ (as in Figure 1) yields $y_t \boldsymbol{u}^\top \hat{\boldsymbol{x}}_t - y_t r_t = \frac{1}{2}||\boldsymbol{u} - \boldsymbol{v}_{k(t-1)}||^2 - \frac{1}{2}||\boldsymbol{u} - \boldsymbol{v}_{k(t)}||^2 + \frac{1}{2}||\boldsymbol{v}_{k(t-1)} - \boldsymbol{v}_{k(t)}||^2$, holding for any $\boldsymbol{u} \in \mathbb{R}^n$. On the other hand, if trial $t$ is such that $M_t Z_t = 0$ we have $\boldsymbol{v}_{k(t-1)} = \boldsymbol{v}_{k(t)}$. Hence we conclude that the equality

$$M_t Z_t \big( y_t \boldsymbol{u}^\top \hat{\boldsymbol{x}}_t - y_t r_t \big) = \frac{1}{2}||\boldsymbol{u} - \boldsymbol{v}_{k(t-1)}||^2 - \frac{1}{2}||\boldsymbol{u} - \boldsymbol{v}_{k(t)}||^2 + \frac{1}{2}||\boldsymbol{v}_{k(t-1)} - \boldsymbol{v}_{k(t)}||^2$$

actually holds for all trials $t$. We sum over $t = 1, \ldots, T$ while observing that $M_t Z_t = 1$ implies both $||\boldsymbol{v}_{k(t-1)} - \boldsymbol{v}_{k(t)}|| = 1$ and $y_t r_t \leq 0$. Recalling that $\boldsymbol{v}_{k(0)} = \boldsymbol{0}$ and rearranging we obtain

$$\sum_{t=1}^T M_t Z_t \big( y_t \boldsymbol{u}^\top \hat{\boldsymbol{x}}_t + |r_t| - \tfrac{1}{2} \big) \leq \tfrac{1}{2}||\boldsymbol{u}||^2, \qquad \forall \boldsymbol{u} \in \mathbb{R}^n. \qquad (1)$$

Now, since the previous inequality holds for any comparison vector $\boldsymbol{u} \in \mathbb{R}^n$, we stretch $\boldsymbol{u}$ to $\frac{b+1/2}{\gamma} \boldsymbol{u}$, being $\gamma > 0$ a free parameter. Then, by the very definition of $D_\gamma(\boldsymbol{u}; (\hat{\boldsymbol{x}}_t, y_t))$, $\frac{b+1/2}{\gamma} y_t \boldsymbol{u}^\top \hat{\boldsymbol{x}}_t \geq \frac{b+1/2}{\gamma} \big( \gamma - D_\gamma(\boldsymbol{u}; (\hat{\boldsymbol{x}}_t, y_t)) \big) \, \forall \gamma > 0$. Plugging into (1) and rearranging,

$$\sum_{t=1}^T M_t Z_t(b + |r_t|) \leq (b + \tfrac{1}{2}) \sum_{t \in \mathcal{U}_T} \frac{1}{\gamma} D_\gamma(\boldsymbol{u}; (\hat{\boldsymbol{x}}_t, y_t)) + \frac{(2b+1)^2}{8\gamma^2} ||\boldsymbol{u}||^2 . \qquad (2)$$

---

**ALGORITHM** Selective sampling second-order Perceptron algorithm
Parameter $b > 0$.
**Initialization:** $A_0 = I$; $\boldsymbol{v}_0 = \boldsymbol{0}$; $k = 1$.
**For** $t = 1, 2, \ldots$ **do**:
    1. Get $\boldsymbol{x}_t \in \mathbb{R}^n$ and set $r_t = \boldsymbol{v}_{k-1}^\top (A_{k-1} + \hat{\boldsymbol{x}}_t \hat{\boldsymbol{x}}_t^\top)^{-1} \hat{\boldsymbol{x}}_t$, $\hat{\boldsymbol{x}}_t = \boldsymbol{x}_t / \|\boldsymbol{x}_t\|$;
    2. predict with $\hat{y}_t = \text{SGN}(r_t) \in \{-1, +1\}$;
    3. draw a Bernoulli random variable $Z_t \in \{0, 1\}$ of parameter

$$\frac{b}{b + |r_t| + \frac{1}{2} r_t^2 \left(1 + \hat{\boldsymbol{x}}_t^\top A_{k-1}^{-1} \hat{\boldsymbol{x}}_t\right)}; \qquad (3)$$

    4. **if** $Z_t = 1$ **then**:
        (a) ask for label $y_t \in \{-1, +1\}$,
        (b) **if** $\hat{y}_t \neq y_t$ **then** update as follows:
            $\boldsymbol{v}_k = \boldsymbol{v}_{k-1} + y_t \hat{\boldsymbol{x}}_t$, $A_k = A_{k-1} + \hat{\boldsymbol{x}}_t \hat{\boldsymbol{x}}_t^\top$, $k \leftarrow k + 1$.

---

Figure 2: The selective sampling second-order Perceptron algorithm.

From Figure 1 we see that $\mathbb{E}[Z_t \mid Z_1, \ldots, Z_{t-1}] = \frac{b}{b + |r_t|}$. Therefore, taking expectations on both sides of (2),

$$\mathbb{E}\left[\sum_{t=1}^T M_t Z_t (b + |r_t|)\right] = \sum_{t=1}^T \mathbb{E}\left[\mathbb{E}\left[M_t Z_t \left(b + |r_t|\right) \mid Z_1, \ldots, Z_{t-1}\right]\right]$$

$$= \sum_{t=1}^T \mathbb{E}\left[M_t \left(b + |r_t|\right) \mathbb{E}\left[Z_t \mid Z_1, \ldots, Z_{t-1}\right]\right] = \mathbb{E}\left[\sum_{t=1}^T M_t\right] b.$$

Replacing back into (2) and dividing by $b$ proves the claimed bound on $\mathbb{E}\left[\sum_{t=1}^T M_t\right]$. The value of $\mathbb{E}\left[\sum_{t=1}^T Z_t\right]$ (the expected number of queried labels) trivially follows from
$$\mathbb{E}\left[\sum_{t=1}^T Z_t\right] = \mathbb{E}\left[\sum_{t=1}^T \mathbb{E}[Z_t \mid Z_1, \ldots, Z_{t-1}]\right]. \qquad \square$$

We now consider the selective sampling version of the second-order Perceptron algorithm, as defined in [5]. See Figure 2. Unlike the first-order algorithm, the second-order algorithm mantains a vector $\boldsymbol{v} \in \mathbb{R}^n$ and a matrix $A \in \mathbb{R}^n \times \mathbb{R}^n$ (whose initial value is the identity matrix $I$). The algorithm predicts through the sign of the margin quantity $r_t = \boldsymbol{v}_{k-1}^\top (A_{k-1} + \hat{\boldsymbol{x}}_t \hat{\boldsymbol{x}}_t^\top)^{-1} \hat{\boldsymbol{x}}_t$, and decides whether to ask for the label $y_t$ through a randomized rule similar to the one in Figure 1. The analysis follows the same pattern as the proof of Theorem 1. A key step in this analysis is a one-trial progress equation developed in [10] for a regression framework. See also [4]. Again, the comparison between the second-order Perceptron's bound and the one contained in Theorem 2 reveals that the selective sampling algorithm can achieve, in expectation, the same mistake bound (see Remark 1) using fewer labels.

**Theorem 2** *Using the notation of Theorem 1, the expected number of mistakes made by the algorithm in Figure 2 is upper bounded by*

$$\inf_{\gamma > 0} \inf_{\boldsymbol{u} \in \mathbb{R}^n} \left( \mathbb{E}\left[\sum_{t \in \mathcal{U}_T} \frac{1}{\gamma} D_\gamma(\boldsymbol{u}; (\hat{\boldsymbol{x}}_t, y_t))\right] + \frac{b}{2\gamma^2} \boldsymbol{u}^\top \mathbb{E}\left[A_{k(T)}\right] \boldsymbol{u} + \frac{1}{2b} \sum_{i=1}^n \mathbb{E} \ln\left(1 + \lambda_i\right) \right),$$

*where $\lambda_1, \ldots, \lambda_n$ are the eigenvalues of the (random) correlation matrix $\sum_{t \in \mathcal{U}_T} \hat{\boldsymbol{x}}_t \hat{\boldsymbol{x}}_t^\top$ and $A_{k(T)} = I + \sum_{t \in \mathcal{U}_T} \hat{\boldsymbol{x}}_t \hat{\boldsymbol{x}}_t^\top$ (thus $1 + \lambda_i$ is the $i$-th eigenvalue of $A_{k(T)}$). The expected number of labels queried by the algorithm is equal to $\sum_{t=1}^T \mathbb{E}\left[\frac{b}{b + |r_t| + \frac{1}{2} r_t^2 \left(1 + \hat{\boldsymbol{x}}_t^\top A_{k-1}^{-1} \hat{\boldsymbol{x}}_t\right)}\right]$.*

*Proof sketch.* The proof proceeds along the same lines as the proof of Theorem 1. Thus we only emphasize the main differences. In addition to the notation given there, we define

$\mathcal{U}_t$ as the set of update trials up to time $t$, i.e., $\mathcal{U}_t = \{i \le t \ : \ M_i Z_i = 1\}$, and $R_t$ as the (random) function $R_t(\boldsymbol{u}) = \frac{1}{2}||\boldsymbol{u}||^2 + \sum_{i \in \mathcal{U}_t} \frac{1}{2}(y_i - \boldsymbol{u}^\top \hat{\boldsymbol{x}}_i)^2$. When trial $t$ is such that $M_t Z_t = 1$ we can exploit a result contained in [10] for linear regression (proof of Theorem 3 therein), where it is essentially shown that choosing $r_t = \boldsymbol{v}_{k-1}^\top A_{k(t)}^{-1} \hat{\boldsymbol{x}}_t$ (as in Figure 2) yields

$$\tfrac{1}{2}(y_t - r_t)^2 = \inf_{\boldsymbol{u} \in \mathbb{R}^n} R_t(\boldsymbol{u}) - \inf_{\boldsymbol{u} \in \mathbb{R}^n} R_{t-1}(\boldsymbol{u}) + \tfrac{1}{2}\left(\hat{\boldsymbol{x}}_t^\top A_{k(t)}^{-1} \hat{\boldsymbol{x}}_t - r_t^2\, \hat{\boldsymbol{x}}_t^\top A_{k(t)-1}^{-1} \hat{\boldsymbol{x}}_t\right). \quad (4)$$

On the other hand, if trial $t$ is such that $M_t Z_t = 0$ we have $\mathcal{U}_t = \mathcal{U}_{t-1}$, thus $\inf_{\boldsymbol{u} \in \mathbb{R}^n} R_{t-1}(\boldsymbol{u}) = \inf_{\boldsymbol{u} \in \mathbb{R}^n} R_t(\boldsymbol{u})$. Hence the equality

$$\tfrac{1}{2} M_t Z_t \big((y_t - r_t)^2 + r_t^2\, \hat{\boldsymbol{x}}_t^\top A_{k(t)-1}^{-1} \hat{\boldsymbol{x}}_t\big)$$
$$= \inf_{\boldsymbol{u} \in \mathbb{R}^n} R_t(\boldsymbol{u}) - \inf_{\boldsymbol{u} \in \mathbb{R}^n} R_{t-1}(\boldsymbol{u}) + \tfrac{1}{2} M_t Z_t\, \hat{\boldsymbol{x}}_t^\top A_{k(t)}^{-1} \hat{\boldsymbol{x}}_t \quad (5)$$

holds for all trials $t$. We sum over $t = 1, \dots, T$, and observe that by definition $R_T(\boldsymbol{u}) = \frac{1}{2}||\boldsymbol{u}||^2 + \sum_{t=1}^T \frac{M_t Z_t}{2}(y_i - \boldsymbol{u}^\top \hat{\boldsymbol{x}}_i)^2$ and $R_0(\boldsymbol{u}) = \frac{1}{2}||\boldsymbol{u}||^2$ (thus $\inf_{\boldsymbol{u} \in \mathbb{R}^n} R_0(\boldsymbol{u}) = 0$). After some manipulation one can see that (5) implies

$$\sum_{t=1}^T M_t Z_t \big(y_t\, \boldsymbol{u}^\top \hat{\boldsymbol{x}}_t + |r_t| + \tfrac{1}{2} r_t^2 (1 + \hat{\boldsymbol{x}}_t^\top A_{k(t)-1}^{-1} \hat{\boldsymbol{x}}_t)\big)$$
$$\le \tfrac{1}{2}\boldsymbol{u}^\top A_{k(T)} \boldsymbol{u} + \sum_{t=1}^T \tfrac{1}{2} M_t Z_t\, \hat{\boldsymbol{x}}_t^\top A_{k(t)}^{-1} \hat{\boldsymbol{x}}_t, \quad (6)$$

holding for any $\boldsymbol{u} \in \mathbb{R}^n$. We continue by elaborating on (6). First, as in [4, 10, 5], we upper bound the quadratic terms $\hat{\boldsymbol{x}}_t^\top A_{k(t)}^{-1} \hat{\boldsymbol{x}}_t$ by[2] $\ln \frac{\det(A_{k(t)})}{\det(A_{k(t)-1})}$. This gives

$$\sum_{t=1}^T \tfrac{1}{2} M_t Z_t\, \hat{\boldsymbol{x}}_t^\top A_{k(t)}^{-1} \hat{\boldsymbol{x}}_t \le \tfrac{1}{2} \ln \frac{\det(A_{k(T)})}{\det(A_0)} = \tfrac{1}{2} \sum_{i=1}^n \ln (1 + \lambda_i) \ .$$

Second, as in the proof of Theorem 1, we stretch the comparison vector $\boldsymbol{u} \in \mathbb{R}^n$ to $\frac{b}{\gamma}\boldsymbol{u}$ and introduce hinge loss terms. We obtain:

$$\sum_{t=1}^T M_t Z_t \Big(b + |r_t| + \tfrac{1}{2} r_t^2 (1 + \hat{\boldsymbol{x}}_t^\top A_{k(t)-1}^{-1} \hat{\boldsymbol{x}}_t)\Big)$$
$$\le b \sum_{t \in \mathcal{U}_T} \tfrac{1}{\gamma} D_\gamma(\boldsymbol{u}; (\hat{\boldsymbol{x}}_t, y_t)) + \tfrac{b^2}{2\gamma^2}\boldsymbol{u}^\top A_{k(T)} \boldsymbol{u} + \tfrac{1}{2} \sum_{i=1}^n \ln (1 + \lambda_i). \quad (7)$$

The bounds on $\mathbb{E}\left[\sum_{t=1}^T M_t\right]$ and $\mathbb{E}\left[\sum_{t=1}^T Z_t\right]$ can now be obtained by following the proof of Theorem 1. $\qquad\qquad\square$

**Remark 1** *The bounds in Theorems 1 and 2 depend on the choice of parameter $b$. As a matter of fact, the optimal tuning of this parameter is easily computed. Let us set for brevity $\hat{D}_\gamma(\boldsymbol{u}; S) = \mathbb{E}\left[\sum_{t \in \mathcal{U}_T} \tfrac{1}{\gamma} D_\gamma(\boldsymbol{u}; (\hat{\boldsymbol{x}}_t, y_t))\right]$. Choosing[3] $b = \frac{1}{2}\sqrt{1 + \frac{4\gamma^2}{||\boldsymbol{u}||^2}\hat{D}_\gamma(\boldsymbol{u}; S)}$ in Theorem 1 gives the following bound on the expected number of mistakes:*

$$\inf_{\boldsymbol{u} \in \mathbb{R}^n} \left(\hat{D}_\gamma(\boldsymbol{u}; S) + \frac{||\boldsymbol{u}||^2}{2\gamma^2} + \frac{||\boldsymbol{u}||}{2\gamma}\sqrt{\hat{D}_\gamma(\boldsymbol{u}; S) + \frac{||\boldsymbol{u}||^2}{4\gamma^2}}\right) \ . \quad (8)$$

*This is an expectation version of the mistake bound for the standard (first-order) Perceptron algorithm [14]. Notice, that in the special case when the data are linearly separable with margin $\gamma^*$ the optimal tuning simplifies to $b = 1/2$ and yields the familiar Perceptron bound $||\boldsymbol{u}||^2/(\gamma^*)^2$. On the other hand, if we set $b = \gamma\sqrt{\frac{\sum_{i=1}^n \mathbb{E}\ln(1+\lambda_i)}{\boldsymbol{u}^\top \mathbb{E}[A_{k(T)}]\boldsymbol{u}}}$ in Theorem 2 we are led to the bound*

$$\inf_{\boldsymbol{u} \in \mathbb{R}^n} \left(\hat{D}_\gamma(\boldsymbol{u}; S) + \tfrac{1}{\gamma}\sqrt{(\boldsymbol{u}^\top \mathbb{E}\left[A_{k(T)}\right]\boldsymbol{u})\sum_{i=1}^n \mathbb{E}\ln(1 + \lambda_i)}\right) \ , \quad (9)$$

*which is an expectation version of the mistake bound for the (deterministic) second-order Perceptron algorithm, as proven in [5]. As it turns out, (8) and (9) might be even sharper than their deterministic counterparts. In fact, the set of update trials $\mathcal{U}_T$ is on average significantly smaller than the one for the deterministic algorithms. This tends to shrink the three terms $\hat{D}_\gamma(\boldsymbol{u}; S)$, $\boldsymbol{u}^\top \mathbb{E}\left[A_{k(T)}\right] \boldsymbol{u}$, and $\sum_{i=1}^n \mathbb{E} \ln\left(1 + \lambda_i\right)$, the main ingredients of the selective sampling bounds.*

**Remark 2** *Like any Perceptron-like algorithm, the algorithms in Figures 1 and 2 can be efficiently run in any given reproducing kernel Hilbert space (e.g., [9, 21, 23]), just by turning them into equivalent dual forms. This is actually what we did in the experiments reported in the next section.*

## 4  Experiments

The empirical evaluation of our algorithms was carried out on two datasets of free-text documents. The first dataset is made up of the first (in chronological order) $40,000$ newswire stories from Reuters Corpus Volume 1 (RCV1) [2]. The resulting set of examples was classified over 101 categories. The second dataset is a specific subtree of the OHSUMED corpus of medical abstracts [1]: the subtree rooted in "Quality of Health Care" (MeSH code N05.712). From this subtree we randomly selected a subset of $40,000$ abstracts. The resulting number of categories was 94. We performed a standard preprocessing on the datasets – details will be given in the full paper.

Two kinds of experiments were made on each dataset. In the first experiment we compared the selective sampling algorithms in Figures 1 and 2 (for different values of $b$), with the standard second-order Perceptron algorithm (requesting all labels). Such a comparison was devoted to studying the extent to which a reduced number of label requests might lead to performance degradation. In the second experiment, we compared variable vs. constant label-request rate. That is, we fixed a few values for parameter $b$, run the selective sampling algorithm in Figure 2, and computed the fraction of labels requested over the training set. Call this fraction $\hat{p} = \hat{p}(b)$. We then run a second-order selective sampling algorithm with (constant) label request probability equal to $\hat{p}$ (independent of $t$). The aim of this experiment was to investigate the effectiveness of a margin-based selective sampling criterion, as opposed to a random one.

Figure 3 summarizes the results we obtained on RCV1 (the results on OHSUMED turned out to be similar, and are therefore omitted from this paper). For the purpose of this graphical representation, we selected the 50 most frequent categories from RCV1, those with frequency larger than 1%. The standard second-order algorithm is denoted by 2ND-ORDER-ALL-LABELS, the selective sampling algorithms in Figures 1 and 2 are denoted by 1ST-ORDER and 2ND-ORDER, respectively, whereas the second-order algorithm with constant label request is denoted by 2ND-ORDER-FIXED-BIAS.[4] As evinced by Figure 3(a), there is a range of values for parameter $b$ that makes 2ND-ORDER achieve almost the same performance as 2ND-ORDER-ALL-LABELS, but with a substantial reduction in the total number of queried labels.[5] In Figure 3(b) we report the results of running 2ND-ORDER, 1ND-ORDER and 2ND-ORDER-FIXED-BIAS after choosing values for $b$ that make the average F-measure achieved by 2ND-ORDER just slightly larger than those achieved by the other two algorithms. We then compared the resulting label request rates and found 2ND-ORDER largely best among the three algorithms (its instantaneous label rate after $40,000$ examples is less than 19%). We made similar experiments for specific categories in RCV1. On the most frequent ones (such as category 70 – Figure 3(c)) this behavior gets emphasized. Finally, in Figure 3(d) we report a direct macroaveraged F-measure comparison

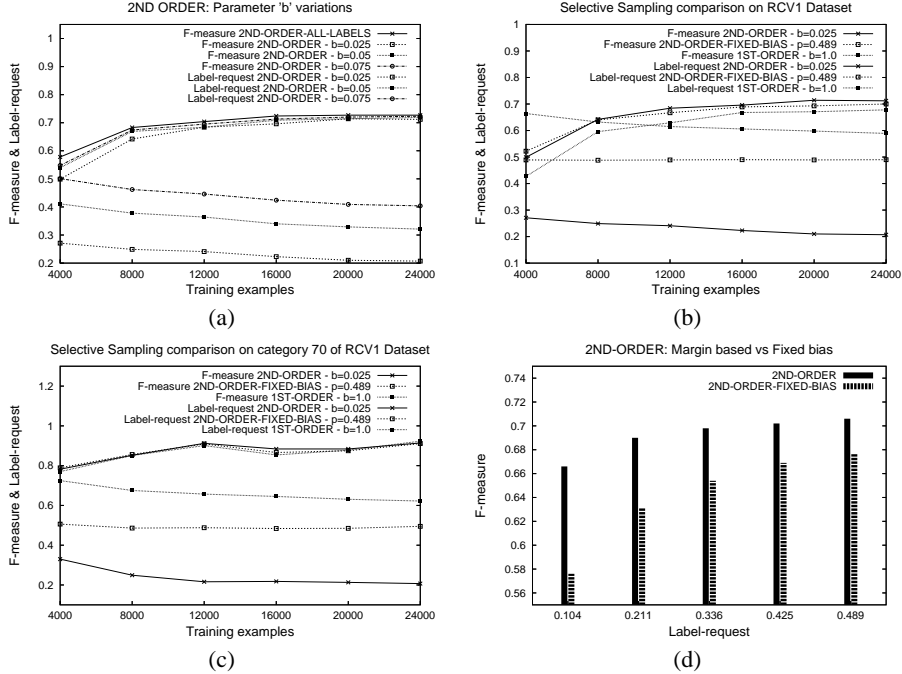

Figure 3: Instantaneous F-measure and instantaneous label-request rate on the RCV1 dataset. We solved a binary classification problem for each class and then (macro)averaged the results. All curves tend to flatten after about $24,000$ examples (out of $40,000$). (a) Instantaneous macroaveraged F-measure of 2ND-ORDER (for three values of $b$) and their corresponding label-request curves. For the very sake of comparison, we also included the F-measure of 2ND-ORDER-ALL-LABELS. (b) Comparison among 2ND-ORDER, 1ST-ORDER and 2ND-ORDER-FIXED-BIAS. (c) Same comparison on a specific category. (d) F-measure of 2ND-ORDER vs. F-measure of 2ND-ORDER-FIXED-BIAS for 5 values of parameter $b$, after $40,000$ examples.

between 2ND-ORDER and 2ND-ORDER-FIXED-BIAS for 5 values of $b$. On the x-axis are the resulting 5 values of the constant bias $\hat{p}(b)$. As expected, 2ND-ORDER outperforms 2ND-ORDER-FIXED-BIAS, though the difference between the two tends to shrink as $b$ (or, equivalently, $\hat{p}(b)$) gets larger.

## 5  Conclusions and open problems

We have introduced new Perceptron-like selective sampling algorithms for learning linear-threshold functions. We analyzed these algorithms in a worst-case on-line learning setting, providing bounds on both the expected number of mistakes and the expected number of labels requested. Our theoretical investigation naturally arises from the traditional way margin-based algorithms are analyzed in the mistake bound model of on-line learning [18, 15, 11, 13, 14, 5]. This investigation suggests that our worst-case selective sampling algorithms can achieve on average the same accuracy as that of their more standard relatives, but allowing a substantial label saving. These theoretical results are corroborated by our empirical comparison on textual data, where we have shown that: (1) the selective sampling algorithms tend to be unaffected by observing less and less labels; (2) if we fix ahead of time the total number of label observations, the margin-driven way of distributing these observations over the training set is largely more effective than a random one.

We close by two simple open questions. (1) Our selective sampling algorithms depend on a scale parameter $b$ having a significant influence on their practical performance. Is there any

principled way of adaptively tuning $b$ so as to reduce the algorithms' sensitivity to tuning parameters? (2) Theorems 1 and 2 do not make any explicit statement about the number of weight updates/support vectors computed by our selective sampling algorithms. We would like to see a theoretical argument that enables us to combine the bound on the number of mistakes with that on the number of labels, giving rise to a meaningful upper bound on the number of updates.

## Footnotes

[1]The cumulative hinge loss measures to what extent hyperplane $\boldsymbol{u}$ separates $S$ at margin $\gamma$. This is also called the *soft margin* in the SVM literature [23, 9, 21].

[2]Here $\det$ denotes the determinant.

[3]Clearly, this tuning relies on information not available ahead of time, since it depends on the whole sequence of examples. The same holds for the choice of $b$ giving rise to (9).

[4]We omitted to report on the first-order algorithms 1ST-ORDER-ALL-LABELS and 1ST-ORDER-FIXED-BIAS, since they are always outperformed by their corresponding second-order algorithms.

[5]Notice that the figures are plotting *instantaneous* label rates, hence the overall fraction of queried labels is obtained by integration.

## References

[1] The OHSUMED test collection. URL: medir.ohsu.edu/pub/ohsumed/.

[2] Reuters corpus volume 1. URL: about.reuters.com/researchandstandards/corpus/.

[3] Atlas, L., Cohn, R., and Ladner, R. (1990). Training connectionist networks with queries and selective sampling. In *NIPS 2*. MIT Press.

[4] Azoury, K.S., and Warmuth, M.K. (2001). Relative loss bounds for on-line density estimation with the exponential familiy of distributions. *Machine Learning*, 43(3):211–246, 2001.

[5] Cesa-Bianchi, N., Conconi, A., and Gentile, C. (2002). A second-order Perceptron algorithm. In *Proc. 15th COLT*, pp. 121–137. LNAI 2375, Springer.

[6] Cesa-Bianchi, N. Lugosi, G., and Stoltz, G. (2004). Minimizing Regret with Label Efficient Prediction In *Proc. 17th COLT*, to appear.

[7] Cesa-Bianchi, N., Conconi, A., and Gentile, C. (2003). Learning probabilistic linear-threshold classifiers via selective sampling. In *Proc. 16th COLT*, pp. 373–386. LNAI 2777, Springer.

[8] Campbell, C., Cristianini, N., and Smola, A. (2000). Query learning with large margin classifiers. In Proc. *17th ICML*, pp. 111–118. Morgan Kaufmann.

[9] Cristianini, N., and Shawe-Taylor, J. (2001). *An Introduction to Support Vector Machines*. Cambridge University Press.

[10] Forster, J. On relative loss bounds in generalized linear regression. (1999). In *Proc. 12th Int. Symp. FCT*, pp. 269–280, Springer.

[11] Freund, Y., and Schapire, R. E. (1999). Large margin classification using the perceptron algorithm. *Machine Learning*, 37(3), 277–296.

[12] Freund, Y., Seung, S., Shamir, E., and Tishby, N. (1997). Selective sampling using the query by committee algorithm. *Machine Learning*, 28(2/3):133–168.

[13] Gentile, C. & Warmuth, M. (1998). Linear hinge loss and average margin. In *NIPS 10*, MIT Press, pp. 225–231.

[14] Gentile, C. (2003). The robustness of the $p$-norm algorithms. *Machine Learning*, 53(3), 265–299.

[15] Grove, A.J., Littlestone, N., & Schuurmans, D. (2001). General convergence results for linear discriminant updates. *Machine Learning*, 43(3), 173–210.

[16] Helmbold, D.P., Littlestone, N. and Long, P.M. (2000). Apple tasting. *Information and Computation*, 161(2), 85–139.

[17] Helmbold, D.P., and Panizza, S. (1997). Some label efficient learning results. In *Proc. 10th COLT*, pp. 218–230. ACM Press.

[18] Littlestone, N. (1988). Learning quickly when irrelevant attributes abound: a new linear-threshold algorithm. *Machine Learning*, 2(4), 285–318.

[19] Littlestone, N., and Warmuth, M.K. (1994). The weighted majority algorithm. *Information and Computation*, 108(2), 212–261.

[20] F. Rosenblatt. (1958). The Perceptron: A probabilistic model for information storage and organization in the brain. *Psychol. Review*, 65, 386–408.

[21] Schölkopf, B., and Smola, A. (2002). *Learning with kernels*. MIT Press, 2002.

[22] Tong, S., and Koller, D. (2000). Support vector machine active learning with applications to text classification. In *Proc. 17th ICML*. Morgan Kaufmann.

[23] Vapnik, V.N. (1998). *Statistical Learning Theory*. Wiley.

[24] Vovk, V. (1990). Aggregating strategies. *Proc. 3rd COLT*, pp. 371–383. Morgan Kaufmann.
